# The VC-Dimension versus the Statistical Capacity of Multilayer Networks

Chuanyi Ji *and Demetri Psaltis
Department of Electrical Engineering
California Institute of Technology
Pasadena, CA 91125

## Abstract

A general relationship is developed between the VC-dimension and the statistical lower epsilon-capacity which shows that the VC-dimension can be lower bounded (in order) by the statistical lower epsilon-capacity of a network trained with random samples. This relationship explains quantitatively how generalization takes place after memorization, and relates the concept of generalization (consistency) with the capacity of the optimal classifier over a class of classifiers with the same structure and the capacity of the Bayesian classifier. Furthermore, it provides a general methodology to evaluate a lower bound for the VC-dimension of feedforward multilayer neural networks.

This general methodology is applied to two types of networks which are important for hardware implementations: two layer $(N - 2L - 1)$ networks with binary weights, integer thresholds for the hidden units and zero threshold for the output unit, and a single neuron $((N - 1)$ networks) with binary weigths and a zero threshold. Specifically, we obtain $O(\frac{W}{lnL}) \leq d_2 \leq O(W)$, and $d_1 \sim O(N)$. Here $W$ is the total number of weights of the $(N - 2L - 1)$ networks. $d_1$ and $d_2$ represent the VC-dimensions for the $(N - 1)$ and $(N - 2L - 1)$ networks respectively.

## 1 Introduction

The information capacity and the VC-dimension are two important quantities that characterize multilayer feedforward neural networks. The former characterizes their

memorization capability, while the latter represents the sample complexity needed for generalization. Discovering their relationships is of importance for obtaining a better understanding of the fundamental properties of multilayer networks in learning and generalization.

In this work we show that the VC-dimension of feedforward multilayer neural networks, which is a distribution-and network-parameter-indenpent quantity, can be lower bounded (in order) by the statistical lower epsilon-capacity $C_\epsilon^-$ (McEliece et.al, (1987)), which is a distribution-and network-dependent quantity, when the samples are drawn from two classes: $\Omega_1(+1)$ and $\Omega_2(-1)$. The only requirement on the distribution from which samples are drawn is that the optimal classification error achievable, the Bayes error $P_{be}$, is greater than zero. Then we will show that the VC-dimension $d$ and the statistical lower epsilon-capacity $C_\epsilon^-$ are related by

$$C_\epsilon^- \leq Ad, \tag{1}$$

where $\epsilon = P_{eo} - \epsilon'$ for $0 < \epsilon' \leq P_{eo}$; or $\epsilon = P_{be} - \epsilon'$ for $0 < \epsilon' \leq P_{be}$. Here $\epsilon$ is the error tolerance, and $P_{eo}$ represents the optimal error rate achievable on the class of classifiers considered. It is obvious that $P_{eo} \geq P_{be}$. The relation given in Equation (1) is non-trivial if $P_{be} > 0$, $P_{eo} \leq \epsilon'$ or $P_{be} \leq \epsilon'$ so that $\epsilon$ is a non-negative quantity. $Ad$ is called the universal sample bound for generalization, where $A < \frac{128ln\frac{1}{\epsilon}}{\epsilon'^2}$ is a positive constant. When the sample complexity exceeds $Ad$, all the networks of the same architechture for all distributions of the samples can generalize with almost probability 1 for $d$ large. A special case of interest, in which $P_{be} = \frac{1}{2}$, corresponds to random assignments of samples. Then $C_\epsilon^-$ represents the random storage capacity which characterizes the memorizing capability of networks.

Although the VC-dimension is a key parameter in generalization, there exists no systematic way of finding it. The relationship we have obtained, however, brings concomitantly a constructive method of finding a lower bound for the VC-dimension of multilayer networks. That is, if the weights of a network are properly constructed using random samples drawn from a chosen distribution, the statistical lower epsilon-capacity can be evaluated and then utilized as bounds for the VC-dimension. In this paper we will show how this constructive approach contributes to finding lower bounds of the VC-dimension of multilayer networks with binary weights.

## 2  A Relationship Between the VC-Dimension and the Statistical Capacity

### 2.1  Definition of the Statistical Capacity

Consider a network $s$ whose weights are constructed from $M$ random samples belonging to two classes. Let $\hat{r}(s) = \frac{Z}{M}$, where $Z$ is the total number of samples classified incorrectly by the network $s$. Then the random variable $\hat{r}(s)$ is the training error rate. Let

$$P_\epsilon(M) = \Pr(\hat{r}(s) \leq \epsilon), \tag{2}$$

where $0 \leq \epsilon \leq 1$. Then the statistical lower epsilon-capacity (statistical capacity in short) $C_\epsilon^-$ is the maximum $M$ such that $P_\epsilon(M) \geq 1 - \eta$, where $\eta$ can be arbitrarily small for sufficiently large $N$.

Roughly speaking, the statistical lower epsilon-capacity defined here can be regarded as a sharp transition point on the curve $P_\epsilon(M)$ shown in Fig.1. When the number of samples used is below this sharp transition, the network can memorize them perfectly.

## 2.2    The Universal Sample Bound for Generalization

Let $P_e(x|s)$ be the true probability of error for the network $s$. Then the generalization error $\Delta E(s)$ satisfies $\Delta E(s) =| \hat{r}(s) - P_e(x|s) |$. We can show that the probability for the generalization error to exceed a given small quantity $\epsilon$ satisfies the following relation.

**Theorem 1**

$$\Pr(\max_{s \in S}\Delta E(s) > \epsilon^{'}) \leq h(2M; d, \epsilon'), \qquad (3)$$

*where*

$$h(2M; d, \epsilon') = \begin{cases} 1; & \textit{either } 2M \leq d, \textit{ or } 6\frac{(2M)^d}{d!}e^{-\frac{\epsilon'^2 M}{8}} \geq 1 \& 2M > d, \\ 6\frac{(2M)^d}{d!}e^{-\frac{\epsilon'^2 M}{8}}; & \textit{otherwise.} \end{cases}$$

*Here $S$ is a class of networks with the same architecture. The function $h(2M; d, \epsilon')$ has one sharp transition occurring at $Ad$ shown in Fig.1, where $A$ is a constant satisfying the equation $A = ln(2A) + 1 - \frac{\epsilon'^2}{8}A = 0$.*

This theorem says that when the number $M$ of samples used exceeds $Ad$, generalization happens with probability 1. Since $Ad$ is a distribution-and-network-parameter-independent quantity, we call it the universal sample bound for generalization.

## 2.3    A Relationship between The VC-Dimension and $C_\epsilon^-$

Roughly speaking, since both the statistical capacity and the VC-dimension represent sharp transition points, it is natural to ask whether they are related. The relationship can actually be given through the theorem below.

**Theorem 2** *Let samples belonging to two classes $\Omega_1(+1)$ and $\Omega_2(-1)$ be drawn independently from some distribution. The only requirement on the distributions considered is that the Bayes error $P_{be}$ satisfies $0 < P_{be} \leq \frac{1}{2}$. Let $S$ be a class of feedforward multilayer networks with a fixed structure consisting of threshold elements and $s_1$ be one network in $S$, where the weights of $s_1$ are constructed from $M$ (training) samples drawn from one distribution as specified above. For a given distribution, let $P_{eo}$ be the optimal error rate achievable on $S$ and $P_{be}$ be the Bayes error rate. Then*

$$\Pr(\hat{r}(s_1) < P_{eo} - \epsilon^{'}) \leq h(2M; d, \epsilon'), \qquad (4)$$

*and*

$$\Pr(\hat{r}(s_1) < P_{be} - \epsilon^{'}) \leq h(2M; d, \epsilon'), \qquad (5)$$

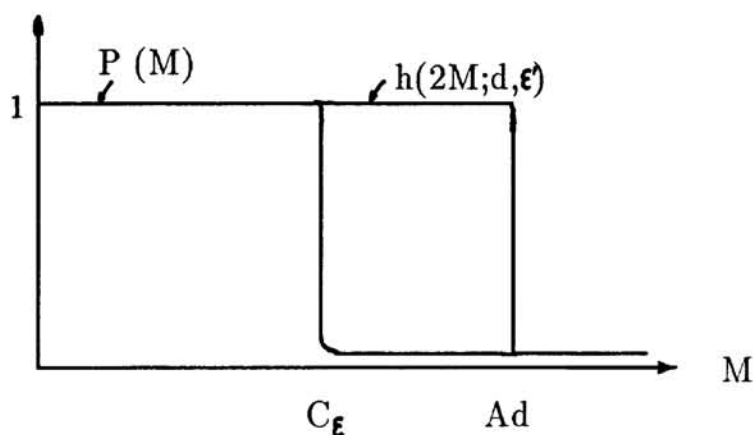

Figure 1: Two sharp transition points for the capacity and the universal sample bound for generalization.

where $\hat{r}(s_1)$ is equal to the training error rate of $s_1$. (It is also called the resubstitution error estimator in the pattern recognition literature.) These relations are nontrivial if $P_{eo} > \epsilon'$, $P_{be} > \epsilon'$ and $\epsilon' > 0$ small.

The key idea of this result is illustrated in Fig.1. That is, the sharp transition which stands for the lower epsilon-capacity is below the sharp transition for the universal sample bound for generalization.

To interpret this relation, let us compare Equation (2) and Equation (5) and examine the range of $\epsilon$ and $\epsilon'$ respectively. Since $\epsilon'$, which is initially given in Inequality (3), represents a bound on the generalization error, it is usually quite small. For most of practical problems, $P_{be}$ is small also. If the structure of the class of networks is properly chosen so that $P_{eo} \approx P_{be}$, then $\epsilon = P_{eo} - \epsilon'$ will be a small quantity. Although the epsilon-capacity is a valid quantity depending on $M$ for any network in the class, for $M$ sufficiently large, the meaningful networks to be considered through this relation is only a small subset in the class whose true probability of error is close to $P_{eo}$. That is, this small subset contains only those networks which can approximate the best classifier contained in this class.

For a special case in which samples are assigned randomly to two classes with equal probability, we have a result stated in Corollary 1.

**Corollary 1** *Let samples be drawn independently from some distribution and then assigned randomly to two classes $\Omega_1(+1)$ and $\Omega_2(-1)$ with equal probability. This is equivalent to the case that the two class conditional distributions have complete overlap with one another. That is, $\Pr(x \mid \Omega_1) = \Pr(x \mid \Omega_2)$. Then the Bayes error is $\frac{1}{2}$. Using the same notation as in the above theorem, we have*

$$C_{\frac{1}{2}-\epsilon'}^{-} \leq Ad. \tag{6}$$

Although the distributions specified here give an uninteresting case for classification purposes, we will see later that the random statistical epsilon-capacity in Inequality (6) can be used to characterize the memorizing capability of networks, and to formulate a constructive approach to find a lower bound for the VC-dimension.

# 3   Bounds for the VC-Dimension of Two Networks with Binary Weights

## 3.1   A Constructive Methodology

One of the applications of this relation is that it provides a general constructive approach to find a lower bound for the VC-dimension for a class of networks. Specifically, using the relationship given in Inequality (6), the procedures can be described as follows.

1) Select a distribution.

2) Draw samples independently from the chosen distribution, and then assign them randomly to two classes.

3) Evaluate the lower epsilon-capacity and then use it as a lower bound for the VC-dimension.

Two example are given below to demonstrate how this general approach can be applied to find lower bounds for the VC-dimension.

## 3.2   Bounds for Two-Layer Networks with Binary Weigths

Two-layer $(N - 2L - 1)$ networks with binary weights and integer thresholds are considered in this section.

### 3.2.1   A lower Bound

The construction of the network we consider is motivated by the one used by Baum (Baum, 1988) in finding the capacity for two layer networks with *real* weights. Although this particular network will fail if the accuracy of the weights and the thresholds is reduced, the idea of using the grandmother-cell type of network will be adopted to construct our network.

We consider a two layer binary network with $2L$ hidden threshold units and one output threshold unit shown in Fig.2 a).

The weights at the second layer are fixed and equal to $+1$ and $-1$ alternately. The hidden units are allowed to have integer thresholds in $[-N, N]$, and the threshold for the output unit is zero.

Let $\vec{X}_l^{(m)} = (x_{l1}^{(m)}, ..., x_{lN}^{(m)})$ be a $N$ dimensional random vector, where $x_{li}^{(m)}$'s are independent random variables taking $(+1)$ and $(-1)$ with equal probability $\frac{1}{2}$, $0 \leq l \leq L$, and $0 \leq m \leq M$. Consider the $l$th pair of hidden units. The weights at the first layer for this pair of hidden units are equal. Let $w_{li}$ denote the weight from the $i$th input to these two hidden units, then we have

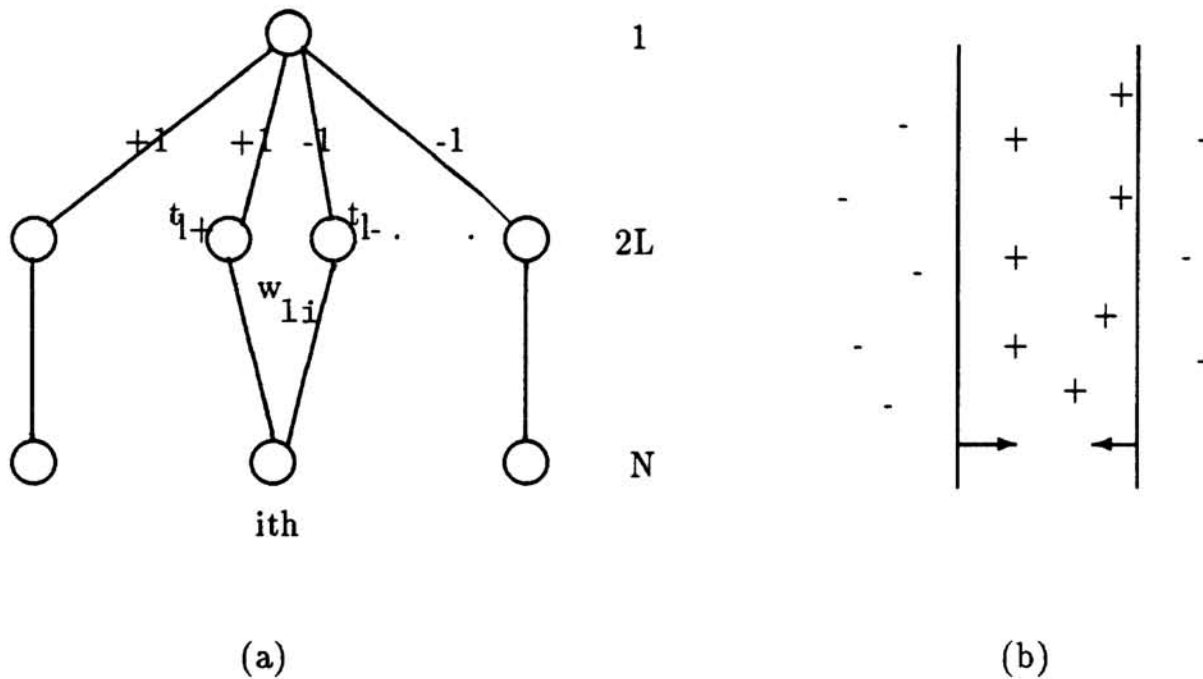

(a)                                                             (b)

Figure 2: a) The two-layer network with binary weights. b) Illustration on how a pair of hidden units separates samples.

$$w_{li} = sgn(\alpha_l \sum_{m=1}^{M} x_{li}^{(m)}),\qquad(7)$$

where $sgn(x) = 1$ if $x > 0$, and $-1$ otherwise. $\alpha_l$'s , $1 \leq l \leq L$, which are independent random variables which take on two values $+1$ or $-1$ with equal probability, represent the random assignments of the $LM$ samples into two classes $\Omega_1(+1)$ and $\Omega_2(-1)$.

The thresholds for these two units are different and are given as

$$t_{l\pm} = \alpha_l \lfloor (1 \mp k)\sqrt{\frac{2}{\pi}} \frac{N}{\sqrt{M}} \rfloor,\qquad(8)$$

where $0 < k < 1$, and $t_{l\pm}$ correspond to the thresholds for the units with weight $+1$ and $-1$ at the second layer respectively.

Fig.2 b) illustrates how this network works. Each pair of hidden units forms two parallel hyperplanes separated by the two thresholds, which will generates a presynaptic input either $+2$ or $(-2)$ to the output unit only for the samples stored in this pair which fall in between the planes when $\alpha_l$ equals either $+1$ or $-1$, and a presynaptic input 0 for the samples falling outside. When the samples as well as the parallel hyperplanes are random, with a certain probability they will fall either between a pair of parallel hyperplanes or outside. Therefore, statistical analysis is needed to obtain the lower epsilon-capacity.

**Theorem 3** *A lower bound $C_{\frac{1}{2}-\epsilon'}^{-'}$ for the lower epsilon-capacity $C_{\frac{1}{2}-\epsilon'}^{-}$ for this network is:*

$$
\begin{aligned}
C_{\frac{1}{2}-\epsilon'}^{-'} &= \frac{(1-k)^2 NL}{\pi(ln\frac{4\sqrt{\pi}\epsilon'^2}{9}L - \frac{1}{2}lnlnL)} \\
&\sim O(\frac{W}{lnL}).
\end{aligned}
\tag{9}
$$

### 3.2.2   An Upper Bound

Since the total number of possible mappings of two layer $(N-2L-1)$ networks with binary weights and integer thresholds ranging in $[-N, N]$ is bounded by $2^{W+L\log 2N}$, the VC-dimension $d_2$ is upper bounded by $W + L\log 2N$, which is in the order of $W$. Then $d_2 \leq O(W)$. By combining both the upper and lower bounds, we have

$$
O(\frac{W}{lnL}) \leq d_2 \leq O(W).
\tag{10}
$$

### 3.3   Bounds for One-Layer Networks with Binary Weigths

The one-layer network we consider here is equivalent to one hidden unit in the above $(N-2L-1)$ network. Specifically, the weight from the $i$-th input unit to the neuron is

$$
w_i = sgn(\sum_{m=1}^{M} \alpha_m x_i^{(m)}),
\tag{11}
$$

where $(1 \leq i \leq N)$, $x_i^{(m)}$'s and $\alpha_m$'s are independent and equally probable binary$(\pm 1)$ random variables, which represent elements of N-dimensional sample vectors and their random assignments to two classes respectively.

**Theorem 4** *The lower epsilon-capacity $C_{\frac{1}{2}-\epsilon'}^{-}$ of this network satisfies*

$$
C_{\frac{1}{2}-\epsilon'}^{-} \approx \frac{N}{\pi^2 \epsilon^2}.
\tag{12}
$$

Then by Corollary 1 we have $O(N) \leq O(d_1)$, where $d_1$ is the VC-dimension of one-layer $(N-1)$ networks.

Using the similar counting arguement, an upper bound can be obtained as $d_1 \leq N$. Then combining the lower and upper bounds, we have $d_1 \sim O(N)$

## 4   Discussions

The general relationship we have drawn between the VC-dimension and the statistical lower epsilon-capacity provides a new view on the sample complexity for generalization. Specifically, it has two implications to learning and generalziation.

1) For random assignments of the samples $(P_{be} = \frac{1}{2})$, the relationship confirms that generalization occurs after memorization, since the statistical lower epsilon-capacity

for this case is the random storage capacity which charaterizes the memorizing capability of networks and it is upper bounded by the universal sample bound for generalization.

2) For cases where the Bayes error is smaller than $\frac{1}{2}$, the relationship indicates that an appropriate choice of a network structure is very important. If a network structure is properly chosen so that the optimal achievable error rate $P_{eo}$ is close to the Bayes error $P_{eb}$, than the optimal network in this class is the one which has the largest lower epsilon-capacity. Since a suitable structure can hardly be chosen a priori due to the lack of knowledge about the underlying distribution, searching for network structures as well as weight values becomes necessary. Similar idea has been addressed by Devroye (Devroye, 1988) and by Vapnik (Vapnik, 1982) for structural minimization.

We have applied this relation as a general constructive approach to obtain lower bounds for the VC-dimension of two-layer and one-layer networks with binary interconnections. For the one-layer networks, the lower bound is tight and matches the upper bound. For the two-layer networks, the lower bound is smaller than the upper bound (in order) by a *ln* factor. In an independent work by Littlestone (Littlestone, 1988), the VC-dimension of so-called DNF expressions were obtained. Since any DNF expression can be implemented by a two layer network of threshold units with binary weights and integer thresholds, this result is equivalent to showing that the VC-dimension of such networks is $O(W)$. We believe that the *ln* factor in our lower bound is due to the limitations of the grandmother-cell type of networks used in our construction.

## Acknowledgement

The authors would like to thank Yaser Abu-Mostafa and David Haussler for helpful discussions. The support of AFOSR and DARPA is gratefully acknowledged.

## Footnotes

*Present Address: Department of Electrical Computer and System Engineering, Rensselaer Polytech Institute, Troy, NY 12180.

## References

E. Baum. (1988) On the Capacity of Multilayer Perceptron. *J. of Complexity*, 4:193-215.

L. Devroye. (1988) Automatic Pattern Recognition: A Study of Probability of Error. *IEEE Trans. on Pattern Recognition and Machine Intelligence*, Vol. 10, No.4: 530-543.

N. Littlestone. (1988) Learning Quickly When Irrelevant Attributes Abound: A New Linear-Threshold Algorithm. *Machine Learning 2:* 285-318.

R.J . McEliece, E.C . Posner, E.R . Rodemich, S.S . Venkatesh. (1987) The Capacity of the Hopfield Associative Memory. *IEEE Trans. Inform. Theory*, Vol. IT-33, No. 4, 461-482.

V.N . Vapnik (1982) *Estimation of Dependences Based on Empirical Data*, New York: Springer-Verlag.
